# An Empirical Analysis of Domain Adaptation Algorithms for Genomic Sequence Analysis

**Gabriele Schweikert**[1]
Max Planck Institutes
Spemannstr. 35-39, 72070 Tübingen, Germany
Gabriele.Schweikert@tue.mpg.de

**Christian Widmer**[1]
Friedrich Miescher Laboratory
Spemannstr. 39, 72070 Tübingen, Germany
ZBIT, Tübingen University
Sand 14, 72076 Tübingen, Germany
Christian.Widmer@tue.mpg.de

**Bernhard Schölkopf**
Max Planck Institute for biol. Cybernetics
Spemannstr. 38, 72070 Tübingen, Germany
Bernhard.Schoelkopf@tue.mpg.de

**Gunnar Rätsch**
Friedrich Miescher Laboratory
Spemannstr. 39, 72070 Tübingen, Germany
Gunnar.Raetsch@tue.mpg.de

## Abstract

We study the problem of domain transfer for a supervised classification task in mRNA splicing. We consider a number of recent domain transfer methods from machine learning, including some that are novel, and evaluate them on genomic sequence data from model organisms of varying evolutionary distance. We find that in cases where the organisms are not closely related, the use of domain adaptation methods can help improve classification performance.

## 1 Introduction

Ten years ago, an eight-year lasting collaborative effort resulted in the first completely sequenced genome of a multi-cellular organism, the free-living nematode *Caenorhabditis elegans*. Today, a decade after the accomplishment of this landmark, 23 eukaryotic genomes have been completed and more than 400 are underway. The genomic sequence builds the basis for a large body of research on understanding the biochemical processes in these organisms. Typically, the more closely related the organisms are, the more similar the biochemical processes. It is the hope of biological research that by analyzing a wide spectrum of model organisms, one can approach an understanding of the full biological complexity. For some organisms, certain biochemical experiments can be performed more readily than for others, facilitating the analysis of particular processes. This understanding can then be transferred to other organisms, for instance by verifying or refining models of the processes—at a fraction of the original cost. This is but one example of a situation where transfer of knowledge across domains is fruitful.

In machine learning, the above information transfer is called *domain adaptation*, where one aims to use data or a model of a well-analyzed *source domain* to obtain or refine a model for a less analyzed *target domain*. For supervised classification, this corresponds to the case where there are ample labeled examples $(\mathbf{x}_i, y_i), i = 1, \ldots, m$ for the source domain, but only few such examples $(\mathbf{x}_i, y_i), i = m + 1, \ldots, m + n$ for the target domain ($n \ll m$). The examples are assumed to be drawn independently from the joint probability distributions $P_S(X, Y)$ and $P_T(X, Y)$, respectively. The distributions $P_S(X, Y) = P_S(Y|X) \cdot P_S(X)$ and $P_T(X, Y) = P_T(Y|X) \cdot P_T(X)$ can differ in several ways:

**(1)** In the classical *covariate shift* case, it is assumed that only the distributions of the input features $P(X)$ varies between the two domains: $P_S(X) \neq P_T(X)$. The conditional, however, remains

invariant, $P_S(Y|X) = P_T(Y|X)$. For a given feature vector $\mathbf{x}$ the label $y$ is thus independent of the domain from which the example stems. An example thereof would be if a function of some biological material is conserved between two organisms, but its composition has changed (e.g. a part of a chromosome has been duplicated).

**(2)** In a more difficult scenario the conditionals differ between domains, $P_S(Y|X) \neq P_T(Y|X)$, while $P(X)$ may or may not vary. This is the more common case in biology. Here, two organisms may have evolved from a common ancestor and a certain biological function may have changed due to evolutionary pressures. The evolutionary distance may be a good indicator for how well the function is conserved. If this distance is small, we have reason to believe that the conditionals may not be completely different, and knowledge of one of them should then provide us with some information also about the other one.

While such knowledge transfer is crucial for biology, and performed by biologists on a daily basis, surprisingly little work has been done to exploit it using machine learning methods on biological databases. The present paper attempts to fill this gap by studying a realistic biological domain transfer problem, taking into account several of the relevant dimensions in a common experimental framework:

- methods — over the last years, the field of machine learning has seen a strong increase in interest in the domain adaptation problem, reflected for instance by a recent NIPS workshop

- domain distance — ranging from close organisms, where simply combining training sets does the job, to distant organisms where more sophisticated methods can potentially show their strengths

- data set sizes — whether or not it is worth transferring knowledge from a distant organism is expected to depend on the amount of data available for the target system

With the above in mind, we selected the problem of mRNA splicing (see Figure A1 in the Appendix for more details) to assay the above dimensions of domain adaptation on a task which is relevant to modern biology. The paper is organized as follows: In Section 2, we will describe the experimental design including the datasets, the underlying classification model, and the model selection and evaluation procedure. In Section 3 we will briefly review a number of known algorithms for domain adaptation, and propose certain variations. In Section 4 we show the results of our comparison with a brief discussion.

## 2 Experimental Design

### 2.1 A Family of Classification Problems

We consider the task of identifying so-called acceptor splice sites within a large set of potential splice sites based on a sequence window around a site. The idea is to consider the recognition of splice sites in different organisms: In all cases, we used the very well studied model organism *C. elegans* as the source domain. As target organisms we chose two additional nematodes, namely, the close relative *C. remanei*, which diverged from *C. elegans* 100 million years ago [10], and the more distantly related *P. pacificus*, a lineage which has diverged from *C. elegans* more than 200 million years ago [7]. As a third target organism we used *D. melanogaster*, which is separated from *C. elegans* by 990 million years [11]. Finally, we consider the plant *A. thaliana*, which has diverged from the other organisms more than 1,600 million years ago. It is assumed that a larger evolutionary distance will likely also have led to an accumulation of functional differences in the molecular splicing machinery. We therefore expect that the differences of classification functions for recognizing splice sites in these organisms will increase with increasing evolutionary distance.

### 2.2 The Classification Model

It has been demonstrated that Support Vector Machines (SVMs) [1] are well suited for the task of splice site predictions across a wide range of organisms [9]. In this work, the so-called *Weighted Degree* kernel has been used to measure the similarity between two example sequences $\mathbf{x}$ and $\mathbf{x}'$ of

fixed length $L$ by counting co-occurring substrings in both sequences at the same position:

$$k_\ell^{\text{wd}}(\mathbf{x}, \mathbf{x}') \quad = \quad \frac{1}{L} \sum_{l=1}^{L-l+1} \sum_{d=1}^{\ell} \beta_d \mathbf{I}\left(\mathbf{x}_{[l:l+d]} = \mathbf{x}'_{[l:l+d]}\right) \qquad (1)$$

where $\mathbf{x}_{[l:l+d]}$ is the substring of length $d$ of $\mathbf{x}$ at position $l$ and $\beta_d = 2\frac{\ell-d+1}{\ell^2+\ell}$ is the weighting of the substring lengths.

In our previous study we have used sequences of length $L = 140$ and substrings of length $\ell = 22$ for splice site detection [9]. With the four-letter DNA sequence alphabet $\{A, C, G, T\}$ this leads to a very high dimensional feature space ($> 10^{13}$ dimensions). Moreover, to archive the best classification performance, a large number of training examples is very helpful ([9] used up to 10 million examples).

For the designed experimental comparison we had to run all algorithms many times for different training set sizes, organisms and model parameters. We chose the source and target training set as large as possible–in our case at most 100,000 examples per domain. Moreover, not for all algorithms we had efficient implementations available that can make use of kernels. Hence, in order to perform this study and to obtain comparable results, we had to restrict ourselves to a case were we can explicitly work in the feature space, if necessary (i.e. $\ell$ not much larger than two). We chose $\ell = 1$. Note, that this choice does not limit the generality of this study, as there is no strong reason, why efficient implementations that employ kernels could not be developed for all methods. The development of large scale methods, however, was not the main focus of this study.

Note that the above choices required an equivalent of about 1500 days of computing time on state-of-the-art CPU cores. We therefore refrained from including more methods, examples or dimensions.

### 2.3 Splits and Model Selection

In the first set of experiments we randomly selected a source dataset of 100,000 examples from *C. elegans*, while data sets of sizes 2,500, 6,500, 16,000, 40,000 and 100,000 were selected for each target organism. Subsequently we performed a second set of experiments where we combined several sources. For our comparison we used 25,000 labeled examples from each of four remaining organisms to predict on a target organism. We ensured that the positives to negatives ratio is at 1/100 for all datasets. Two thirds of each target set were used for training, while one third was used for evaluation in the course of hyper-parameter tuning.[1] Additionally, test sets of 60,000 examples were set aside for each target organism. All experiments were repeated three times with different training splits (source and target), except the last one which always used the full data set. Reported will be the average area under the precision-recall-curve (auPRC) and its standard deviation, which is considered a sensible measure for imbalanced classification problems. The data and additional information will be made available for download on a supplementary website.[2]

## 3 Methods for Domain Adaptation

Regarding the distributional view that was presented in Section 1, the problem of splice site prediction can be affected by both evils simultaneously, namely $P_S(X) \neq P_T(X)$ and $P_S(Y|X) \neq P_T(Y|X)$, which is also the most realistic scenario in the case of modeling most biological processes. In this paper, we will therefore drop the classical covariate shift assumption, and allow for different predictive functions $P_S(Y|X) \neq P_T(Y|X)$.

### 3.1 Baseline Methods ($SVM_S$ and $SVM_T$)

As baseline methods for the comparison we consider two methods: (a) training on the source data only ($\text{SVM}_S$) and (b) training on the target data only ($\text{SVM}_T$). For $\text{SVM}_S$ we use the source data for training however we tune the hyper-parameter on the available target data. For $\text{SVM}_T$ we use the available target data for training (67%) and model selection (33%). The resulting functions are

$$f_S(\mathbf{x}) = \langle \Phi(\mathbf{x}), \mathbf{w}_S \rangle + b_S \qquad \text{and} \qquad f_T(\mathbf{x}) = \langle \Phi(\mathbf{x}), \mathbf{w}_T \rangle + b_T.$$

### 3.2 Convex Combination ($SVM_S$+$SVM_T$)

The most straightforward idea for domain adaptation is to reuse the two optimal functions $f_T$ and $f_S$ as generated by the base line methods $SVM_S$ and $SVM_T$ and combine them in a convex manner:

$$F(\mathbf{x}) = \alpha f_T(\mathbf{x}) + (1 - \alpha) f_S(\mathbf{x}).$$

Here, $\alpha \in [0, 1]$ is the convex combination parameter that is tuned on the evaluation set (33%) of the target domain. A great benefit of this approach is its efficiency.

### 3.3 Weighted Combination ($SVM_{S+T}$)

Another simple idea is to train the method on the union of source and target data. The relative importance of each domain is integrated into the loss term of the SVM and can be adjusted by setting domain-dependent cost parameters $C_S$ and $C_T$ for the $m$ and $n$ training examples from the source and target domain, respectively:

$$\min_{\mathbf{w},\xi} \quad \frac{1}{2}\|\mathbf{w}\|^2 + C_S \sum_{i=1}^{m} \xi_i + C_T \sum_{i=m+1}^{m+n} \xi_i \tag{2}$$

$$\text{s.t.} \quad y_i(\langle \mathbf{w}, \Phi(\mathbf{x}_i)\rangle + b) \geq 1 - \xi_i \quad \forall i \in [1, m+n]$$
$$\xi_i \geq 0 \quad \forall i \in [1, m+n]$$

This method has two model parameters and requires training on the union of the training sets. Since the computation time of most classification methods increases super-linearly and full model selection may require to train many parameter combinations, this approach is computationally quite demanding.

### 3.4 Dual-task Learning ($SVM_{S,T}$)

One way of extending the weighted combination approach is a variant of multi-task learning [2]. The idea is to solve the source and target classification problems simultaneously and couple the two solutions via a regularization term. This idea can be realized by the following optimization problem:

$$\min_{\mathbf{w}_S,\mathbf{w}_T,\xi} \quad \frac{1}{2}\|\mathbf{w}_S - \mathbf{w}_T\|^2 + C \sum_{i=1}^{m+n} \xi_i \tag{3}$$

$$\text{s.t.} \quad y_i(\langle \mathbf{w}_S, \Phi(\mathbf{x}_i)\rangle + b) \geq 1 - \xi_i \quad \forall i \in 1,\ldots,m$$
$$y_i(\langle \mathbf{w}_T, \Phi(\mathbf{x}_i)\rangle + b) \geq 1 - \xi_i \quad \forall i \in m+1,\ldots,m+n$$
$$\xi_i \geq 0 \quad \forall i \in 1,\ldots,m+n$$

Please note that now $\mathbf{w}_S$ *and* $\mathbf{w}_T$ are optimized. The above optimization problem can be solved using a standard QP-solver. In a preliminary experiment we used the optimization package CPLEX to solve this problem, which took too long as the number of variables is relatively large. Hence, we decided to approximate the soft-margin loss using the logistic loss $l(f(\mathbf{x}), y) = \log(1 + \exp(-yf(\mathbf{x})))$ and to use a conjugate gradient method[3] to minimize the resulting objective function in terms of $\mathbf{w}_S$ and $\mathbf{w}_T$.

### 3.5 Kernel Mean Matching ($SVM_{S \to T}$)

Kernel methods map the data into a reproducing kernel Hilbert space (RKHS) by means of a mapping $\Phi : \mathcal{X} \to \mathcal{H}$ related to a positive definite kernel via $k(\mathbf{x}, \mathbf{x}') = \langle \Phi(\mathbf{x}), \Phi(\mathbf{x}')\rangle$. Depending on the choice of kernel, the space of $\mathcal{H}$ may be spanned by a large number of higher order features of the data. In such cases, higher order statistics for a set of input points can be computed in $\mathcal{H}$ by simply taking the mean (i.e., the first order statistics). In fact, it turns out that for a certain class of kernels, the mapping

$$\mu : (\mathbf{x}_1,\ldots,\mathbf{x}_n) \mapsto \frac{1}{n}\sum_{i=1}^{n}\Phi(\mathbf{x}_i)$$

is injective [5] — in other words, given knowledge of (only) the mean (the right hand side), we can completely reconstruct the set of points. For a characterization of this class of kernels, see for instance [4]. It is often not necessary to retain all information (indeed, it may be useful to specify which information we want to retain and which one we want to disregard, see [8]). Generally speaking, the higher dimensional $\mathcal{H}$, the more information is contained in the mean.

In [6] it was proposed that one could use this for covariate shift adaptation, moving the mean of a source distribution (over the inputs only) towards the mean of a target distribution by re-weighting the source training points. We have applied this to our problem, but found that a variant of this approach performed better. In this variant, we do not re-weight the source points, but rather we translate each point towards the mean of the target inputs:

$$\hat{\Phi}(\mathbf{x}_j) = \Phi(\mathbf{x}_j) - \alpha \left( \frac{1}{m} \sum_{i=1}^{m} \Phi(\mathbf{x}_i) - \frac{1}{n} \sum_{i=m+1}^{m+n} \Phi(\mathbf{x}_i) \right) \qquad \forall j = 1, \ldots, m.$$

This also leads to a modified source input distribution which is statistically more similar to the target distribution and which can thus be used to improve performance when training the target task. Unlike [6], we do have a certain amount of labels also for the target distribution. We make use of them by performing the shift separately for each class $y \in \{\pm 1\}$:

$$\hat{\Phi}(\mathbf{x}_j) = \Phi(\mathbf{x}_j) - \alpha \left( \frac{1}{m_y} \sum_{i=1}^{m} [\![y_i = y]\!] \Phi(\mathbf{x}_i) - \frac{1}{n_y} \sum_{i=m+1}^{m+n} [\![y_i = y]\!] \Phi(\mathbf{x}_i) \right)$$

for all $j = m + 1, \ldots, m + n$ with $y_j = y$, where $m_y$ and $n_y$ are the number of source and target examples with label $y$, respectively. The shifted examples can now be used in different ways to obtain a final classifier. We decided to use the weighted combination with $C_S = C_T$ for comparison.

### 3.6 Feature Augmentation ($SVM_{S \times T}$)

In [3] a method was proposed that augments the features of source and target examples in a domain-specific way:

$$\begin{aligned} \hat{\Phi}(\mathbf{x}) &= (\Phi(\mathbf{x}), \Phi(\mathbf{x}), \mathbf{0})^\top & \text{for } i = 1, \ldots, m \\ \hat{\Phi}(\mathbf{x}) &= (\Phi(\mathbf{x}), \mathbf{0}, \Phi(\mathbf{x}))^\top & \text{for } i = m + 1, \ldots, m + n. \end{aligned}$$

The intuition behind this idea is that there exist one set of parameters that models the properties common to both sets and two additional sets of parameters that model the specifics of the two domains. It can easily be seen that the kernel for the augmented feature space can be computed as:

$$k_{AUG}(\mathbf{x}_i, \mathbf{x}_i) = \begin{cases} 2\langle \Phi(\mathbf{x}_i), \Phi(\mathbf{x}_j) \rangle & \text{if } [\![i \leq m]\!] = [\![j \leq m]\!] \\ \langle \Phi(\mathbf{x}_i), \Phi(\mathbf{x}_j) \rangle & \text{otherwise} \end{cases}$$

This means that the "similarity" between two examples is two times as high, if the examples were drawn from the same domain, as if they were drawn from different domains. Instead of the factor 2, we used a hyper-parameter $B$ in the following.

### 3.7 Combination of Several Sources

Most of the above algorithms can be extended in one way or another to integrate several source domains. In this work we consider only three possible algorithms: (a) convex combinations of several domains, (b) KMM on several domains and (c) an extension of the dual-task learning approach to multi-task learning. We briefly describe these methods below:

**Multiple Convex Combinations ($M$-$SVM_S$+$SVM_T$)** The most general version would be to optimize all convex combination coefficients independently. If done in a grid-search-like manner, it becomes prohibitive for more than say three source domains. In principle, one can optimize these coefficients also by solving a linear program. In preliminary experiments we tried both approaches and they typically did not lead to better results than the following combination:

$$F(\mathbf{x}) = \alpha f_T(\mathbf{x}) + (1 - \alpha) \frac{1}{|\mathcal{S}|} \sum_{S \in \mathcal{S}} f_S(\mathbf{x}),$$

where $\mathcal{S}$ is the set of all considered source domains. We therefore only considered this way of combining the predictions.

**Multiple KMM (*M-SVM$_{S \to T}$*)**   Here, we shift the source examples of each domain independently towards the target examples, but by the same relative distance ($\alpha$). Then we train one classifier on the shifted source examples as well as the target examples.

**Multi-task Learning (*M-SVM$_{S,T}$*)**   We consider the following version of multi-task learning:

$$\min_{\{\mathbf{w}_D\}_{D \in \mathcal{D}}, \xi} \quad \frac{1}{2} \sum_{D_1 \in \mathcal{D}} \sum_{D_2 \in \mathcal{D}} \gamma_{D_1, D_2} \|\mathbf{w}_{D_1} - \mathbf{w}_{D_2}\|^2 + \sum_i \xi_i \tag{4}$$

$$\text{s.t.} \quad y_i (\langle \mathbf{w}_{D_j}, \Phi(\mathbf{x}_i) \rangle + b) \geq 1 - \xi_i \tag{5}$$

$$\xi_i \geq 0$$

for all examples $(\mathbf{x}_i, y_i)$ in domain $D_j \in \mathcal{D}$, where $\mathcal{D}$ is the set of all considered domains. $\gamma$ is a set of regularization parameters, which we parametrized by two parameters $C_S$ and $C_T$ in the following way: $\gamma_{D_1, D_2} = C_S$ if $D_1$ and $D_2$ are source domains and $C_T$ otherwise.

# 4   Experimental Results

We considered two different settings for the comparison. For the first experiment we assume that there is *one* source domain with enough data that should be used to improve the performance in the target domain. In the second setting we analyze whether one can benefit from several source domains.

## 4.1   Single Source Domain

Due to space constraints, we restrict ourselves to presenting a summary of our results with a focus on best and worst performing methods. The detailed results are given in Figure A2 in the appendix, where we show the median auPRC of the methods SVM$_T$, SVM$_S$, SVM$_{S \to T}$, SVM$_{S+T}$, SVM$_S$+SVM$_T$, SVM$_{S \times T}$ and SVM$_{S,T}$ for the considered tasks. The summary is given in Figure 1, where we illustrate which method performed best (green), similarly well (within a confidence interval of $\sigma/\sqrt{n}$) as the best (light green), considerably worse than the best (yellow), not significantly better than the worst (light red) or worst (red). From these results we can make the following observations:

1. Independent of the task, if there is very little target data available, the training on source data performs much better than training on the target data. Conversely, if there is much target data available then training on it easily outperforms training the source data.

2. For a larger evolutionary distance of the target organisms to source organism *C. elegans*, a relatively small number of target training examples for the SVM$_T$ approach is sufficient to achieve similar performance to the SVM$_S$ approach, which is always trained on 100,000 examples. We call the number of target examples with equal source and target performance the break-even point. For instance, for the closely related organism *C. remanei* one needs nearly as many target data as source data to achieve the same performance. For the most distantly related organism *A. thaliana*, less than 10% target data is sufficient to outperform the source model.

3. In almost all cases, the performance of domain adaption algorithms is considerably higher than source (SVM$_S$) and target only (SVM$_T$). This is most pronounced near the break-even point, e.g. 3% improvement for *C. remanei* and 14% for *D. melanogaster*.

4. Among the domain adaptation algorithms, the dual-task learning approach (SVM$_{S,T}$) performed most often best (12/20 cases). Second most often best (5/20) performed the convex combination approach (SVM$_S$+SVM$_T$).

From our observations we can conclude that the simple convex combination approach works surprisingly well. It is only outperformed by the dual-task learning algorithm which performs consistently well for all organisms and target training set sizes.

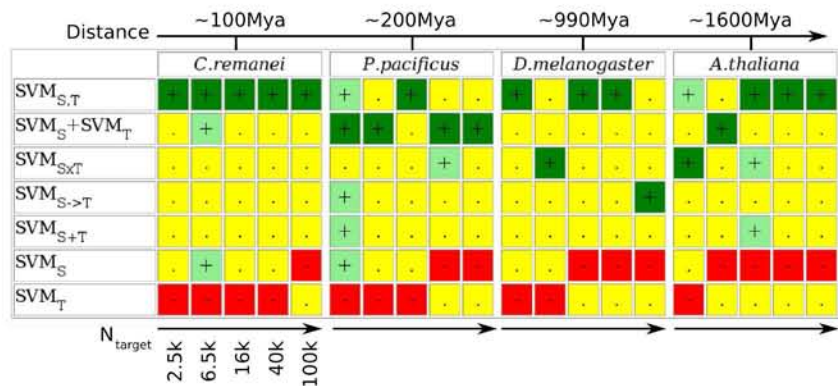

Figure 1: Summary of determined performances for each presented method. Each column contains 5 sub-columns, which correspond to ascending target data set sizes (2,500, 6,500, 16,000, 40,000, 100,000). The method with the highest auPRC score for a given organism and target data set size is assigned a dark green background. Methods that do not perform significantly worse (within a confidence interval of $\sigma/\sqrt{n}$) are shown in light green. Accordingly, the worst performer is shown in red. The remaining methods, that do not fall in any of the mentioned categories are shown in yellow. This figure summarizes the methods based on the single-source domain approach.

## 4.2 Multiple Sources

In a second set of experiments we considered the three algorithms combining several sources. The results are given in Figure A2 in the appendix and a summary in Figure 2. We can make the following observations:

1. Relative to the single source algorithms, these algorithms perform worse, if the additional source organisms are further away from the target organism than the source used by the single source algorithm. For instance, for *C. remanei* this is expected as fewer training examples of the closely related *C. elegans* organism are available.

2. For distantly related target organisms, such as *D. melanogaster* and *A. thaliana* the usage of multiple sources can lead to drastic improvements relative to the single source algorithms, in particular for small target training set sizes. This is in particular noteworthy in the case of *A. thaliana*, where all four source organisms have similar distance to the target organism. We can therefore conclude that a more general model can be learnt from different source organisms of similar distance as compared to a single source organism.

3. Among the multiple source algorithms, convex combinations and multi-task learning are the most successful ones. The first leads to the best results in 4/20 case and the latter in 11/20 cases.

From these observations we can conclude that it pays off to use diverse multiple sources, if there is no very close relative available. The multi-task learning algorithm outperforms the other methods for distantly related organisms.

## 5 Conclusions

Using the example of splice site prediction, we can show that domain adaptation algorithms are well suited to considerably boost predictive performance on sequence-based classification problems. Based on the observations from our experiments, we can conclude that all domain adaptation algorithms lead to considerable improvements over the source and target models. The improvement over baseline methods is most pronounced if the source organism is only distantly related to the target organism. Even though many of the presented domain adaptation methods performed similarly well, we could determine the dual-task algorithm and multi-task learning as the best performing methods in our comparison.

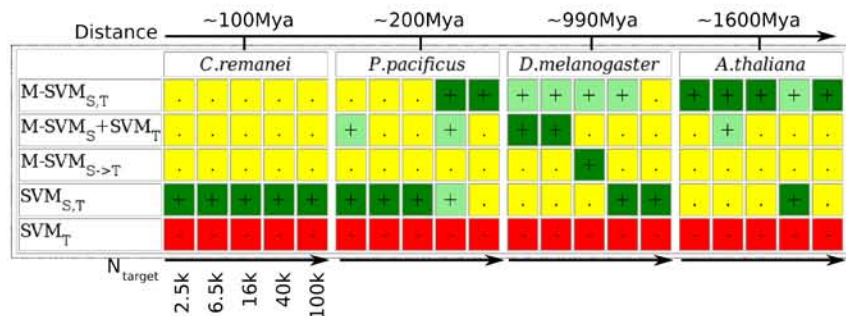

Figure 2: Summary of the performances of multi-source domain methods following the color scheme defined in Figure 1. Briefly, ascending target data set sizes are contained in the sub-columns, the best performing method is shown in dark green, the worst performing method is shown in red, while the remaining colors denote performances in between. For comparison we also included the baseline method $SVM_T$ and the dual task method $SVM_{S,T}$ as the best single source method.

To our knowledge, this work constitutes the first thorough experimental comparison of available domain adaptation algorithms in a well controlled experimental framework. Furthermore, this is one of the first applications of such methods to problems in computational biology, a field that provides a wealth of problems that could greatly benefit from these techniques.

**Acknowledgments** We would like to thank Alexander Zien, Cheng Soon Ong, Petra Philips, Sören Sonnenburg and Arthur Gretton for inspiring discussions and helpful comments on the design of this work. Moreover, we would like to thank Alexander Zien and Carl Rasmussen for providing code used in the comparison.

## Footnotes

[1]These authors contributed equally.

[1]Details on the hyper-parameter settings and tuning are shown in Table A2 in the appendix.

[2]http://www.fml.mpg.de/raetsch/projects/genomedomainadaptation

[3]We used Carl Rasmussen's `minimize` function.

# References

[1] B. Boser, I. Guyon, and V. Vapnik. A training algorithm for optimal margin classifiers. In *Proc. COLT '92*, pages 144–152, Pittsburgh, Pennsylvania, United States, 1992. ACM Press.

[2] R. Caruana. Multitask learning. *Machine Learning*, 28:41–75, 1997.

[3] H. Daume III. Frustratingly easy domain adaptation. In *Conference of the Association for Computational Linguistics (ACL)*, Prague, Czech Republic, 2007.

[4] K. Fukumizu, A. Gretton, X. Sun, and B. Schölkopf. Kernel measures of conditional dependence. In J. Platt, D. Koller, Y. Singer, and S. Roweis, editors, *Advances in Neural Information Processing Systems 20*, pages 489–496. MIT Press, Cambridge, MA, 2008.

[5] A. Gretton, K. Borgwardt, M. Rasch, B. Schölkopf, and A. Smola. A kernel method for the two-sample-problem. In *Advances in Neural Information Processing Systems 19*, Cambridge, MA, 2007. MIT Press.

[6] A. Gretton, A. Smola, J. Huang, M. Schmittfull, and B. Schölkopf. Covariate shift by kernel mean matching. In J. Quiñonero-Candela, M. Sugiyama, A. Schwaighofer, and N. D. Lawrence, editors, *Dataset Shift in Machine Learning*. MIT Press, Cambridge, MA, 2008.

[7] A. Pires-DaSilva and R. Sommer. Conservation of the global sex determination gene tra-1 in distantly related nematodes. *GENES & DEVELOPMENT*, 18(10):1198–1208, MAY 15 2004.

[8] L. Song, X. Zhang, A. Smola, A. Gretton, and B. Schölkopf. Tailoring density estimation via reproducing kernel moment matching. In *Proc. Intl. Conf. Machine Learning*, 2008.

[9] S. Sonnenburg, G. Schweikert, P. Philips, J. Behr, and G. Rätsch. Accurate splice site prediction using support vector machines. *BMC Bioinformatics*, 8(Suppl. 10):S7, 2007.

[10] L. Stein, Z. Bao, D. Blasiar, T. Blumenthal, M. Brent, N. Chen, A. Chinwalla, L. Clarke, C. Clee, A. Coghlan, A. Coulson, P. D'Eustachio, D. Fitch, L. Fulton, R. Fulton, S. Griffiths-Jones, T. Harris, L. Hillier, R. Kamath, P. Kuwabara, E. Mardis, M. Marra, T. Miner, P. Minx, J. Mullikin, R. Plumb, J. Rogers, J. Schein, M. Sohrmann, J. Spieth, J. Stajich, C. Wei, D. Willey, R. Wilson, R. Durbin, and R. Waterston. The genome sequence of Caenorhabditis briggsae: A platform for comparative genomics. *PLOS BIOLOGY*, 1(2):166+, NOV 2003.

[11] A. Ureta-Vidal, L. Ettwiller, and E. Birney. Comparative genomics: Genome-wide analysis in metazoan eukaryotes. *NATURE REVIEWS GENETICS*, 4(4):251–262, APR 2003.
